# Learning Statistically Neutral Tasks without Expert Guidance

**Ton Weijters**
Information Technology,
Eindhoven University,
The Netherlands

**Antal van den Bosch**
ILK,
Tilburg University,
The Netherlands

**Eric Postma**
Computer Science,
Universiteit Maastricht,
The Netherlands

## Abstract

In this paper, we question the necessity of levels of expert-guided abstraction in learning hard, statistically neutral classification tasks. We focus on two tasks, date calculation and parity-12, that are claimed to require intermediate levels of abstraction that must be defined by a human expert. We challenge this claim by demonstrating empirically that a single hidden-layer BP-SOM network can learn both tasks without guidance. Moreover, we analyze the network's solution for the parity-12 task and show that its solution makes use of an elegant intermediary checksum computation.

## 1 Introduction

Breaking up a complex task into many smaller and simpler subtasks facilitates its solution. Such task decomposition has proved to be a successful technique in developing algorithms and in building theories of cognition. In their study and modeling of the human problem-solving process, Newell and Simon [1] employed protocol analysis to determine the subtasks human subjects employ in solving a complex task. Even nowadays, many cognitive scientists take task decomposition, i.e., the necessity of explicit levels of abstraction, as a fundamental property of human problem solving. Dennis Norris' [2] modeling study on the problem-solving capacity of autistic savants is a case in point. In the study, Norris focuses on the date-calculation task (i.e., to calculate the day of the week a given date fell on), which some autistic savants have been reported to perform flawlessly [3]. In an attempt to train a multi-layer neural network on the task, Norris failed to get a satisfactory level of generalization performance. Only by decomposing the task into three sub-tasks, and training the separate networks on each of the sub-tasks, the date-calculation task could be learned. Norris concluded that the date-calculation task is solvable (learnable) only when it is decomposed into intermediary steps using human assistance [2].

The date-calculation task is a very hard task for inductive learning algorithms, because it is a *statistically neutral* task: all conditional output probabilities on any input feature have chance values. Solving the task implies decomposing it, if possible, into subtasks that are not statistically neutral. The only suggested decomposition of the date-calculation task known to date involves explicit assistance

Figure 1: An example BP-SOM network.

from a human supervisor [2]. This paper challenges the decomposition assumption by showing that the date-calculation task can be learned in a single step with a appropriately constrained single hidden-layer neural network. In addition, another statistically neutral task, called the parity-$n$ task (given an $n$-length bit string of 1's and 0's, calculate whether the number of 1's is even or odd) is investigated. In an experimental study by Dehaene, Bossini, and Giraux [4], it is claimed that humans decompose the parity-$n$ task by first counting over the input string, and then perform the even/odd decision. In our study, parity-12 is shown to be learnable by a network with a single hidden layer.

## 2   BP-SOM

Below we give a brief characterization of the functioning of BP-SOM. For details we refer to [5]. The aim of the BP-SOM learning algorithm is to establish a cooperation between BP learning and SOM learning in order to find adequately constrained hidden-layer representations for learning classification tasks. To achieve this aim, the traditional MFN architecture [6] is combined with SOMs [7]: each hidden layer of the MFN is associated with one SOM (See Figure 1). During training of the weights in the MFN, the corresponding SOM is trained on the hidden-unit activation patterns.

After a number of training cycles of BP-SOM learning, each SOM develops a two-dimensional representation, that is translated into classification information, i.e., each SOM element is provided with a class label (one of the output classes of the task). For example, let the BP-SOM network displayed in Figure 1 be trained on a classification task which maps instances to either output class A or B. Three types of elements can be distinguished in the SOM: elements labelled with class A, elements labelled with class B, and unlabelled elements (no winning class could be found). The two-dimensional representation of the SOM is used as an addition to the standard BP learning rule [6]. Classification and reliability information from the SOMs is included when updating the connection weights of the MFN. The error of a hidden-layer vector is an accumulation of the error computed by the BP learning rule, and a SOM-error. The SOM-error is the difference between the hidden-unit activation vector and the vector of its best-matching element associated with the same class on the SOM.

An important effect of including SOM information in the error signals is that clusters of hidden-unit activation vectors of instances associated with the same class tend to become *increasingly similar* to each other. On top of this effect, individual hidden-unit activations tend to become more streamlined, and often end up having activations near one of a limited number of discrete values.

## 3   The date-calculation task

The first statistically neutral calculation task we consider is the date-calculation task: determining the day of the week on which a given date fell. (For instance, *October 24, 1997* fell on a Friday.) Solving the task requires an algorithmic approach that is typically hard for human calculators and requires one or more intermediate steps. It is generally assumed that the identity of these intermediate steps follows from the algorithmic solution, although variations exist in the steps as reportedly used by human experts [2]. We will show that such explicit abstraction is not needed, after reviewing the case for the necessity of "human assistance" in learning the task.

### 3.1   Date calculation with expert-based abstraction

Norris [2] attempted to model autistic savant date calculators using a multi-layer feedforward network (MFN) and the back-propagation learning rule [6]. He intended to build a model mimicking the behavior of the autistic savant without the need either to develop arithmetical skills or to encode explicit knowledge about regularities in the structure of dates. A standard multilayer network trained with backpropagation [6] was not able to solve the date-calculation task. Although the network was able to learn the examples used for training, it did not manage to generalize to novel date-day combinations. In a second attempt Norris split up the date-calculation task in three simpler subtasks and networks.

Using the three-stage learning strategy Norris obtained a nearly perfect performance on the training material and a performance of over 90% on the test material (errors are almost exclusively made on dates falling in January or February in leap years). He concludes with the observation that "The only reason that the network was able to learn so well was because it had some human assistance." [2, p.285]. In addition, Norris claims that "even if the [backpropagation] net did have the right number of layers there would be no way for the net to distribute its learning throughout the net such that each layer learned the appropriate step in computation." [2, p. 290].

### 3.2   Date calculation without expert-based abstraction

We demonstrate that with the BP-SOM learning rule, a single hidden-layer feedforward network can become a successful date calculator. Our experiment compares three types of learning: standard backpropagation learning (BP, [6]), backpropagation learning with weight decay (BPWD, [8]), and BP-SOM learning. Norris used BP learning in his experiment which leads to overfitting [2] (a considerably lower generalization accuracy on new material as compared to reproduction accuracy on training material); BPWD learning was included to avoid overfitting.

The parameter values for BP (including the number of hidden units for each task) were optimized by performing pilot experiments with BP. The optimal learning-rate and momentum values were 0.15 and 0.4, respectively. BP, BPWD, and BP-SOM were trained for a fixed number of cycles $m = 2000$. *Early stopping*, a common method to prevent overfitting, was used in all experiments with BP, BPWD, and BP-SOM [9].

In our experiments with BP-SOM, we used the same interval of dates as used by Norris, i.e., training and test dates ranged from *January 1, 1950* to *December 31, 1999*. We generated two training sets, each consisting of 3,653 randomly selected instances, i.e., one-fifth of all dates. We also generated two corresponding test sets and two validation sets (with 1,000 instances each) of new dates within the same 50-year period. In all our experiments, the training set, test set, and validation set

Table 1: Average generalization performances (plus standard deviation, after '±'; averaged over ten experiments) in terms of incorrectly-processed training and test instances, of BP, BPWD, and BP-SOM, trained on the date-calculation task and the parity-12 task.

| | BP: % incorrect | | BPWD: % incorrect | | BP-SOM: % incorrect | |
|---|---|---|---|---|---|---|
| Task | Train | Test | Train | Test | Train | Test |
| date calc. | 20.8 ±5.4 | 28.8 ±7.8 | 1.5 ± 0.3 | 8.8 ±1.4 | 2.9 ±2.0 | 3.3 ±1.9 |
| parity-12 | 14.1 ±18.8 | 27.4 ±16.4 | 21.6 ±24.2 | 22.4 ±18.3 | 5.9 ±10.2 | 6.2 ±10.7 |

had empty intersections. We partitioned the input into three fields, representing the day of the month (31 units), the month (12 units) and the year (50 units). The output is represented by 7 units, one for each day of the week. The MFN contained one hidden layer with 12 hidden units for BP, and 25 hidden units for BPWD and BP-SOM. The SOM of the BP-SOM network contained $12 \times 12$ elements. Each of the three learning types was tested on two different data sets. Five runs with different random weight initializations were performed on each set, yielding ten runs per learning type. The averaged classification errors on the test material are reported in Table 1.

From Table 1 it follows that the average classification error of BP is high: on test instances BP yields a classification error of 28.8%, while the classification error of BP on training instances is 20.8%. Compared to the classification error of BP, the classification errors on both training and test material of BPWD and BP-SOM are much lower. However, BPWD's generalization performance on the test material is considerably worse than its performance on the training material: a clear indication of overfitting. We note in passing that the results of BPWD contrast with Norris' [2] claim that BP is *unable* to learn the date-calculation task when it is not decomposed into subtasks. The inclusion of weight decay in BP is sufficient for a good approximation of the performance results of Norris' decomposed network.

The results in Table 1 also show that the performance of BP-SOM on test material is significantly better than that of BPWD ($t(19)=7.39$, $p<0.001$); BP-SOM has learned the date-calculation task at a level well beyond the average of human date calculators as reported by Norris [2]. In contrast with Norris' pre-structured network, BP-SOM does not rely on expert-based levels of abstraction for learning the date-calculation task.

## 4   The parity-12 task

The parity-$n$ problem, starting from the XOR problem (parity-2), continues to be a relevant topic on the agenda of many neural network and machine learning researchers. Its definition is simple (determine whether there is an odd or even number of 1's in an $n$-length bit string of 1's and 0's), but established state-of-the-art algorithms such as C4.5 [10] and backpropagation [6] cannot learn it even with small $n$, i.e., backpropagation fails with $n \geq 4$ [11]. That is, these algorithms are unable to generalize from learning instances of a parity-$n$ task to unseen new instances of the same task. As with date calculation, this is due to the statistical neutrality of the task. The solution of the problem must lie in having some comprehensive overview over *all* input values at an intermediary step before the odd/even decision is made. Indeed, humans appear to follow this strategy [4].

Figure 2: Graphic representation of a 7 × 7 SOM associated with a BP-trained MFN (left) and a BPWD-trained MFN (middle), and a 7 × 7 SOM associated with a BP-SOM network (right), all trained on the parity-12 task.

Analogous to our study of the date-calculation task presented in Section 3, we apply BP, BPWD, and BP-SOM to the parity-$n$ task. We have selected $n$ to be 12. The training set contained 1,000 different instances selected at random out of the set of 4,096 possible bit strings. The test set and the validation set contained 100 new instances each. The hidden layer of the MFN in all three algorithms contained 20 hidden units, and the SOM in BP-SOM contained 7 × 7 elements. The algorithms were run with 10 different random weight initializations. Table 1 displays the classification errors on training instances and test instances.

Analysis of the results shows that BP-SOM performs significantly better than BP and BPWD on test material (t(19)=3.42, p<0.01 and t(19)=2.42, p<0.05, respectively). (The average error of 6.2% made by BP-SOM stems from a single experiment out of the ten performing at chance level, and the remaining nine yielding about 1% error). BP-SOM is able to learn the parity-12 task quite accurately; BP and BPWD fail relatively, which is consistent with other findings [11].

As an additional analysis, we have investigated the differences in hidden unit activations after training with the three learning algorithms. To visualize the differences between the representations developed at the hidden layers of the MFNs trained with BP, BPWD, and BP-SOM, we also trained SOMs with the hidden layer activities of the trained BP and BPWD networks. The left part of Figure 2 visualizes the class labelling of the SOM attached to the BP-trained MFN after training; the middle part visualizes the SOM of the BPWD-trained MFN, and the right part displays the SOM of the BP-SOM network after training on the same material. The SOM of the BP-SOM network is much more organized and clustered than that of the SOMs corresponding with the BP-trained and BPWD-trained MFNs. The reliability values of the elements of all three SOMs are represented by the width of the black and white squares. It can be seen that the overall reliability and the degree of clusteredness of the SOM of the BP-SOM network is considerably higher than that of the SOM of the BP-trained and BPWD-trained MFNs.

## 5 How parity-12 is learned

Given the hardness of the task and the supposed necessity of expert guidance, and given BP-SOM's success in learning parity-12 in contrast, it is relevant to analyze what solution was found in the BP-SOM learning process. In this subsection we provide such an analysis, and show that the trained network performs an elegant checksum calculation at the hidden layer as the intermediary step.

All elements of SOMs of BP-SOM networks trained on the parity-12 task are either the prototype for training instances that are all labeled with the same class, or

Table 2: List of some training instances of the parity-12 task associated with SOM elements (1,1), (2,4), and (3,3) of a trained BP-SOM network.

| SOM (1,1), class=even, reliability 1.0 | | | | | | | | | | | | checksum |
|---|---|---|---|---|---|---|---|---|---|---|---|---|
| in1 | in2 | in3 | in4 | in5 | in6 | in7 | in8 | in9 | in10 | in11 | in12 | checksum |
| 1 | 1 | 0 | 0 | 0 | 0 | 0 | 0 | 0 | 0 | 0 | 0 | -2 |
| 0 | 0 | 1 | 0 | 0 | 0 | 1 | 0 | 1 | 1 | 0 | 0 | -2 |
| 1 | 1 | 0 | 1 | 0 | 0 | 0 | 1 | 0 | 0 | 0 | 0 | -2 |
| . | . | . | | | | | | | | | | |

| SOM (2,4), class=odd, reliability 1.0 | | | | | | | | | | | | checksum |
|---|---|---|---|---|---|---|---|---|---|---|---|---|
| in1 | in2 | in3 | in4 | in5 | in6 | in7 | in8 | in9 | in10 | in11 | in12 | checksum |
| 0 | 1 | 1 | 1 | 1 | 0 | 1 | 1 | 0 | 1 | 0 | 0 | -1 |
| 1 | 1 | 1 | 0 | 1 | 1 | 1 | 0 | 1 | 1 | 0 | 1 | -1 |
| 1 | 0 | 1 | 1 | 0 | 1 | 0 | 1 | 1 | 0 | 1 | 0 | -1 |
| . | . | . | | | | | | | | | | |

| SOM (3,3), class=even, reliability 1.0 | | | | | | | | | | | | checksum |
|---|---|---|---|---|---|---|---|---|---|---|---|---|
| in1 | in2 | in3 | in4 | in5 | in6 | in7 | in8 | in9 | in10 | in11 | in12 | checksum |
| 0 | 0 | 1 | 1 | 0 | 0 | 1 | 1 | 0 | 1 | 0 | 1 | 0 |
| 1 | 1 | 1 | 1 | 1 | 0 | 1 | 0 | 0 | 0 | 1 | 1 | 0 |
| 1 | 0 | 1 | 1 | 1 | 1 | 0 | 1 | 1 | 0 | 0 | 1 | 0 |
| . | . | . | | | | | | | | | | |
| - | - | - | + | + | + | - | - | - | + | + | + | |

prototype of no instances at all. Non-empty elements (the black and white squares in the right part of Figure 2) can thus be seen as containers of homogeneously-labeled subsets of the training set (i.e., fully reliable elements). The first step of our analysis consists of collecting, after training, for each non-empty SOM element all training instances clustered at that SOM element. As an illustration, Table 2 lists some training instances clustered at the SOM elements at coordinates (1,1), (2,4), and (3,3). At first sight the only common property of instances associated with the same SOM element is the class to which they belong; e.g., all instances of SOM element (1,1) are even, all instances of SOM element (2,4) are odd, and all instances of SOM element (3,3) are again even.

The second step of our analysis focuses on the sign of the weights of the connections between input and hidden units. Surprisingly, we find that the connections of each individual input unit to all hidden units have the same sign; each input unit can therefore be labeled with a sign marker (as displayed at the bottom of Table 2). This allows the clustering on the SOM to become interpretable. All weights from input unit 1, 2, 3, 7, 8, and 9 to all units of the hidden layer are negative, all weights from input unit 4, 5, 6, 10, 11, and 12 to all units of the hidden layer are positive. At the hidden layer, this information is gathered as if a *checksum* is computed; each SOM element contains instances that add up to an identical checksum. This can already be seen using only the sign information rather than the specific weights. For instance, all instances clustered at SOM element (1,1) lead to a checksum of -2 when a sum is taken of the product of all input values with all weight signs. Analogously, all instances of cluster (2,4) count up to -1 and the instances of cluster (3,3) to zero. The same regularity is present in the instances of the other SOM elements.

In sum, the BP-SOM solution to the parity-12 task can be interpreted as to transform it at the hidden layer into the mapping of different, approximately discrete, checksums to either class 'even' or 'odd'.

## 6  Conclusions

We have performed two learning experiments in which the BP-SOM learning algorithm was trained on the date-calculation task and on the parity-12 task. Both tasks are hard to learn because they are statistically neutral, but can be learned adequately and without expert guidance by the BP-SOM learning algorithm. The effect of the SOM part in BP-SOM (adequately constrained hidden-layer vectors, reliable clustering of vectors on the SOM, and streamlined hidden-unit activations) clearly contributes to this success.

From the results of the experiments on the date-calculation task, we conclude that Norris' claim that, without human assistance, a backpropagation net would never learn the date-calculation task is inaccurate. While BP with weight decay performs at Norris' target level of accuracy, BP-SOM performs even better. Apparently BP-SOM is able to distribute its learning throughout the net such that the two parts of the network (from input layer to hidden layer, and from hidden layer to output layer) perform the mapping with an appropriate intermediary step.

The parity-12 experiment exemplified that such a discovered intermediary step can be quite elegant; it consists of the computation of a checksum via the connection weights between the input and hidden layers. Unfortunately, a similar elegant simplicity was not found in the connection weights and SOM clustering of the date calculation task; future research will be aimed at developing more generic analyses for trained BP-SOM networks, so that automatically-discovered intermediary steps may be made understandably explicit.

## References

[1]   Newell, A. and Simon, H.A. (1972) *Human problem solving.* Engelwood Cliffs, NJ: Prentice-Hall.

[2]   Norris, D. (1989). How to build a connectionist idiot (savant). *Cognition,* **35,** 277–291.

[3]   Hill, A. L. (1975). An investigation of calendar calculating by an idiot savant. *American Journal of Psychiatry,* **132,** 557–560.

[4]   Dehaene, P., Bossini, S., and Giraux, P. (1993). The mental representation of parity and numerical magnitude. *Journal of Experimental Psychology: General,* **122,** 371–396.

[5]   Weijters, A., Van den Bosch, A., Van den Herik, H. J. (1997). Behavioural Aspects of Combining Backpropagation Learning and Self-organizing Maps. *Connection Science,* **9,** 235–252.

[6]   Rumelhart, D. E., Hinton, G. E., and Williams, R. J. (1986). Learning internal representations by error propagation. In D. E. Rumelhart and J. L. McClelland (Eds.), *Parallel Distributed Processing: Explorations in the Microstructure of Cognition,* volume 1: Foundations (pp. 318–362). Cambridge, MA: The MIT Press.

[7]   Kohonen, T. (1989). *Self-organisation and Associative Memory.* Berlin: Springer Verlag.

[8]   Hinton, G. E. (1986). Learning distributed representations of concepts. In *Proceedings of the Eighth Annual Conference of the Cognitive Science Society,* 1–12. Hillsdale, NJ: Erlbaum.

[9]   Prechelt, L. (1994). *Proben1: A set of neural network benchmark problems and benchmarking rules.* Technical Report 24/94, Fakultät für Informatik, Universität Karlsruhe, Germany.

[10]  Quinlan, J. R. (1993). *C4.5: Programs for Machine Learning.* San Mateo, CA: Morgan Kaufmann.

[11]  Thornton, C. (1996). Parity: the problem that won't go away. In G. McCalla (Ed.), *Proceeding of AI-96,* Toronto, Canada (pp. 362-374). Berlin: Springer Verlag.
